# Q-Clustering

**Mukund Narasimhan**[†]　　　　　　**Nebojsa Jojic**[‡]　　　　　　**Jeff Bilmes**[†]

[†]Dept of Electrical Engineering, University of Washington, Seattle WA
[‡]Microsoft Research, Microsoft Corporation, Redmond WA
{mukundn,bilmes}@ee.washington.edu and jojic@microsoft.com

## Abstract

We show that Queyranne's algorithm for minimizing symmetric submodular functions can be used for clustering with a variety of different objective functions. Two specific criteria that we consider in this paper are the single linkage and the minimum description length criteria. The first criterion tries to maximize the minimum distance between elements of different clusters, and is inherently "discriminative". It is known that optimal clusterings into $k$ clusters for any given $k$ in polynomial time for this criterion can be computed. The second criterion seeks to minimize the description length of the clusters given a probabilistic generative model. We show that the optimal partitioning into 2 clusters, and approximate partitioning (guaranteed to be within a factor of 2 of the the optimal) for more clusters can be computed. To the best of our knowledge, this is the first time that a tractable algorithm for finding the optimal clustering with respect to the MDL criterion for 2 clusters has been given. Besides the optimality result for the MDL criterion, the chief contribution of this paper is to show that the *same algorithm* can be used to optimize a broad class of criteria, and hence can be used for many application specific criterion for which efficient algorithm are not known.

## 1　Introduction

The clustering of data is a problem found in many pattern recognition tasks, often in the guises of unsupervised learning, vector quantization, dimensionality reduction, etc. Formally, the clustering problem can be described as follows. Given a finite set $S$, and a criterion function $J_k$ defined on all partitions of $S$ into $k$ parts, find a partition of $S$ into $k$ parts $\{S_1, S_2, \ldots, S_k\}$ so that $J_k(\{S_1, S_2, \ldots, S_k\})$ is maximized. The number of $k$-clusters for a size $n > k$ data set is roughly $k^n/k!$ [5] so exhaustive search is not an efficient solution. The problem, in fact, is NP-complete for most desirable measures. Broadly speaking there are two classes of criteria for clustering. There are distance based criteria, for which a distance measure is specified between each pair of elements, and the criterion somehow combines either intercluster or intracluster distances into an objective function. The other class of criteria are model based, and for these, a probabilistic (generative) model is specified. There is no universally accepted criterion for clustering. The appropriate criterion is typically application dependent, and therefore, we do not claim that the two criteria considered in this paper are inherently better or more generally applicable than other criteria. However, we can show that for the single-linkage criterion, we can compute the optimal clustering into $k$ parts (for any $k$), and for the MDL criterion, we can compute the optimal clustering into 2 parts using Queyranne's algorithm. More generally, any criterion from a

broad class of criterion can be solved by the *same algorithm*, and this class of criteria is closed under linear combinations. In addition to the theoretical elegance of a single algorithm solving a number of very different criterion, this means that we can optimize (for example) for the sum of single-linkage and MDL criterions (or positively scaled versions thereof). The two criterion we consider are quite different. The first, "discriminative", criterion we consider is the single-linkage criterion. In this case, we are given distances $d(s_1, s_2)$ between all elements $s_1, s_2 \in S$, and we try and find clusters that maximize the minimum distance between elements of different clusters (i.e., maximize the separation of the clusters). This criterion has several advantages. Since we are only comparing distances, the distance measure can be chosen from any ordered set (addition/squaring/multiplication of distances need not be defined as is required for K-means, spectral clustering etc.). Further, this criterion only depends on the rank ordering of the distances, and so is completely insensitive to any monotone transformation of the distances. This gives a lot of flexibility in constructing a distance measure appropriate for an application. For example, it is a very natural candidate when the distance measure is derived from user studies (since users are more likely to be able to provide rankings than exact distances). On the other hand, this criterion is sensitive to outliers and may not be appropriate when there are a large number of outliers in the data set. The kernel based criterion considered in [3] is similar in spirit to this one. However, their algorithm only provides approximate solutions, and the extension to more than 2 clusters is not given. However, since they optimize the distance of the clusters to a hyperplane, it is more appropriate if the clusters are to be classified using a SVM.

The second criterion we consider is "generative" in nature and is based on the Minimum Description Length principle. In this case we are given a (generative) probability model for the elements, and we attempt to find clusters so that describing or encoding the clusters (separately) can be done using as few bits as possible. This is also a very natural criterion - grouping together data items that can be highly compressed translates to grouping elements that share common characteristics. This criterion has also been widely used in the past, though the algorithms given do not guarantee optimal solutions (even for 2 clusters).

Since these criteria seem quite different in nature, it is surprising that the same algorithm can be used to find the *optimal* partitions into two clusters in both cases. The key principle here is the notion of submodularity (and its variants) [1, 2]. We will show that the problem of finding the optimal clusterings minimizing the description length is equivalent to the problem of minimizing a symmetric submodular function, and the problem of maximizing the cluster separation is equivalent to minimizing a symmetric function which, while not submodular, is closely related, and can be minimized by the same algorithm.

## 2   Background and Notation

A *clustering* of a finite set $S$ is a partition $\{S_1, S_2, \ldots, S_k\}$ of $S$. We will call the individual elements of the partition the clusters of the partition. If there are $k$ clusters in the partition, then we say that the partition is a $k$-clustering. Let $\mathcal{C}_k(S)$ be the set of all $k$-clusterings for $1 \leq k \leq |S|$. For the first criterion, we assume we are given a function $d : S \times S \to \mathbb{R}$ that represents the "distance" between objects. Intuitively, we expect that $d(s, t)$ is large when the objects are dissimilar. We will assume that $d(\cdot, \cdot)$ is symmetric, but make no further assumptions. In particular we do not assume that $d(\cdot, \cdot)$ is a metric (Later on in this paper, we will not even assume that $d(s, t)$ is a (real) number, but instead will allow the range of $d$ to be a ordered set ). The distance between sets $T$ and $R$ is often defined to be the smallest distance between elements from these different clusters: $D(R, T) = \min_{r \in R, t \in T} d(r, t)$. The single-linkage criterion tries to maximize this distance, and hence an optimal 2-clustering is in $\arg \max_{\{S_1, S_2\} \in \mathcal{C}_2(S)} D(S_1, S_2)$. We let $\mathcal{O}_k(S)$ be the set of all optimal k-clusterings for $1 \leq k \leq |S|$ with respect to $D(\cdot, \cdot)$. It is known that an algorithm based on the Minimum Spanning Tree can be used to find optimal

clusterings for the single-linkage criterion[8].

For the second criterion, we assume $S$ is a collection of random variables, and for any subset $T = \{s_1, s_2, \ldots, s_m\}$ of $S$, we let $H(T)$ be the entropy of the set of random variables $\{s_1, s_2, \ldots, s_m\}$. Now, the (expected) total cost of encoding or describing the set $T$ is $H(T)$. So a partition $\{S_1, S_2\}$ of $S$ that minimizes the description length (DL) is in

$$\underset{\{S_1,S_2\}\in\mathcal{C}_2(S)}{\arg\min} DL(S_1, S_2) = \underset{\{S_1,S_2\}\in\mathcal{C}_2(S)}{\arg\min} H(S_1) + H(S_2)$$

We will denote by $2^S$ the set of all subsets of $S$. A set function $f : 2^S \to \mathbb{R}$ assigns a (real) number to every subset of $S$. We say that $f$ is *submodular* if $f(A) + f(B) \geq f(A \cup B) + f(A \cap B)$ for every $A, B \subseteq S$. $f$ is symmetric if $f(A) = f(S \setminus A)$. In [1], Queyranne gives a polynomial time algorithm that finds a set $A \in 2^S \setminus \{S, \phi\}$ that minimizes any symmetric submodular set function (specified in the form of an oracle). That is, Queyranne's algorithm finds a non-trivial partition $\{S_1, S \setminus S_1\}$ of $S$ so that $f(S_1) (= f(S \setminus S_1))$ minimizes $f$ over all non-trivial subsets of $S$. The problem of finding non-trivial minimizers of a symmetric submodular function can be thought of a a generalization of the graph-cut problem. For a symmetric set function $f$, we can think of $f(S_1)$ as $f(S_1, S \setminus S_1)$, and if we can extend $f$ to be defined on all pairs of disjoint subsets of $S$, then Rizzi showed in [2] that Queyranne's algorithm works even when $f$ is not submodular, as long as $f$ is monotone and consistent, where $f$ is *monotone* if for $R, T, T' \subseteq S$ with $T' \subseteq T$ and $R \cap T = \phi$ we have $f(R, T') \leq f(R, T)$ and $f$ is *consistent* if $f(A, W \cup B) \geq f(B, A \cup W)$ whenever $A, B, W \subseteq S$ are disjoint sets satisfying $f(A, W) \geq f(B, W)$.

The rest of this paper is organized as follows. In Section 3, we show that Queyranne's algorithm can be used to find the optimal $k$-clustering (for any $k$) in polynomial time for the single-linkage criterion. In Section 4, we give an algorithm for finding the optimal clustering into 2 parts that minimizes the description length. In Section 5, we present some experimental results.

## 3 Single-Linkage: Maximizing the separation between clusters

In this section, we show that Queyranne's algorithm can be used for finding $k$-clusters (for any given $k$) that maximize the separation between elements of different clusters. We do this in two steps. First in Subsection 3.1, we show that Queyranne's algorithm can partition the set $S$ into two parts to maximize the distance between these parts in polynomial time. Then in Subsection 3.2, we show how this subroutine can be used to find *optimal $k$* clusters, also in polynomial time.

### 3.1 Optimal 2-clusterings

In this section, we will show that the function $-D(\cdot, \cdot)$ is monotone and consistent. Therefore, by Rizzi's result, it follows that we can find a 2-clustering $\{S_1, S_2\} = \{S_1, S \setminus S_1\}$ that minimizes $-D(S_1, S_2)$, and hence maximizes $D(S_1, S_2)$.

**Lemma 1.** *If $R \subseteq T$, then $D(U, T) \leq D(U, R)$ (and hence $-D(U, R) \leq -D(U, T)$).*

This would imply that $-D$ is monotone. To see this, observe that

$$D(U, T) = \min_{u \in U, t \in T} d(u, t) = \min\left(\min_{u \in U, r \in R} d(u, r), \min_{u \in U, t \in T \setminus R} d(u, t)\right) \leq D(U, R)$$

**Lemma 2.** *Suppose that $A, B, W$ are disjoint subsets of $S$ and $D(A, W) \leq D(B, W)$. Then $D(A, W \cup B) \leq D(B, A \cup W)$.*

To see this first observe that $D(A, B \cup W) = \min(D(A, B), D(A, W))$ because

$$D(A, W \cup B) = \min_{a \in A, x \in W \cup B} D(a, x) = \min\left(\min_{a \in A, w \in W} D(a, w), \min_{a \in A, b \in B} D(A, b)\right)$$

It follows that $D(A, B \cup W) = \min\left(D(A, B), D(A, W)\right) \leq \min\left(D(A, B), D(B, W)\right)$ $= \min\left(D(B, A), D(B, W)\right) = D(B, A \cup W)$. Therefore, if $-D(A, W) \geq -D(B, W)$, then $-D(A, W \cup B) \geq -D(B, A \cup W)$. Hence $-D(\cdot, \cdot)$ is consistent.

Therefore, $-D(\cdot, \cdot)$ is symmetric, monotone and consistent. Hence it can be minimized using Queyranne's algorithm [2]. Therefore, we have a procedure to compute optimal 2-clusterings. We now extend this to compute optimal $k$-clusterings.

### 3.2   Optimal $k$-clusterings

We start off by extending our objective function for $k$-clusterings in the obvious way. The function $D(R, T)$ can be thought of as defining the *separation* or *margin* between the clusters $R$ and $T$. We can generalize this notion to more than two clusters as follows. Let

$$\text{seperation}(\{S_1, S_2, \ldots, S_k\}) = \min_{i \neq j} D(S_i, S_j) = \min_{\substack{S_i \neq S_j \\ s_i \in S_i, s_j \in S_j}} d(s_i, s_j)$$

Note that $\text{seperation}(\{R, T\}) = D(R, T)$ for a 2-clustering. The function $\text{seperation} : \cup_{k=1}^{|S|} \mathcal{C}_k(S) \to \mathbb{R}$ takes a single clustering as its argument. However, $D(\cdot, \cdot)$ takes two disjoint subsets of $S$ as its arguments the union of which need not be $S$ in general. The margin is the distance between the closest elements of different clusters, and hence we will be interested in finding $k$-clusters that maximize the margin. Therefore, we seek an element in $\mathcal{O}_k(S) = \arg\max_{\{S_1, S_2, \ldots, S_k\} \in \mathcal{C}_k(S)} \text{seperation}(\{S_1, S_2, \ldots, S_k\})$. Let $v_k(S)$ be the margin of an element in $\mathcal{O}_k(S)$. Therefore, $v_k(S)$ is the best possible margin of any $k$-clustering of $S$. An obvious approach to generating optimal $k$-clusterings given a method of generating optimal 2-clusterings is the following. Start off with an optimal 2-clustering $\{S_1, S_2\}$. Then apply the procedure to find 2-clusterings of $S_1$ and $S_2$, and stop when you have enough clusters. There are two potential problems with this approach. First, it is not clear that an optimal $k$-clustering can be a refinement of an optimal 2-clustering. That is, we need to be sure that there is an optimal $k$-clustering in which $S_1$ is the union of some of the clusters, and $S_2$ is the union of the remaining. Second, we need to figure out how many of the clusters $S_1$ is the union of and how many $S_2$ is the union of. In this section, we will show that for any $k \geq 3$, there is always an optimal $k$-clustering that is a refinement of any given optimal 2-clustering. A simple dynamic programming algorithm takes care of the second potential problem.

We begin by establishing some relationships between the separation of clusterings of different sizes. To compare the separation of clusterings with different number of clusters, we can try and merge two of the clusters from the clustering with more clusters. Say that $\mathcal{S} = \{S_1, S_2, \ldots, S_k\} \in \mathcal{C}_k(S)$ is any $k$-clustering of $S$, and $\mathcal{S}'$ is a $(k-1)$-clustering of $S$ obtained by merging two of the clusters (say $S_1$ and $S_2$). Then $\mathcal{S}' = \{S_1 \cup S_2, S_3, \ldots, S_k\} \in \mathcal{C}_{k-1}(S)$.

**Lemma 3.** *Suppose that* $\mathcal{S} = \{S_1, S_2, \ldots, S_k\} \in \mathcal{C}_k(S)$ *and* $\mathcal{S}' = \{S_1 \cup S_2, S_3, \ldots, S_k\} \in \mathcal{C}_{k-1}(S)$. *Then* $\text{seperation}(\mathcal{S}) \leq \text{seperation}(\mathcal{S}')$. *In other words, refining a partition can only reduce the margin.*

Therefore, refining a clustering (i.e., splitting a cluster) can only reduce the separation. An immediate corollary is the following.

**Corollary 4.** *If* $\mathcal{T}_l \in \mathcal{C}_l(S)$ *is a refinement of* $\mathcal{T}_k \in \mathcal{C}_k(S)$ *(for $k < l$) then* $\text{seperation}(\mathcal{T}_l) \leq \text{seperation}(\mathcal{T}_k)$. *It follows that* $v_k(S) \geq v_l(S)$ *if* $1 \leq k < l \leq n$.

*Proof.* It suffices to prove the result for $k = l - 1$. The first assertion follows immediately from Lemma 3. Let $\mathcal{S} \in \mathcal{O}_l(S)$ be an optimal $l$-clustering. Merge any two clusters to get $\mathcal{S}' \in \mathcal{C}_k(S)$. By Lemma 3, $v_k(S) \geq \text{seperation}(\mathcal{S}') \geq \text{seperation}(\mathcal{S}) = v_l(S)$. □

Next, we consider the question of constructing larger partitions (i.e., partitions with more clusters) from smaller partitions. Given two clusterings $\mathcal{S} = \{S_1, S_2, \ldots, S_k\} \in \mathcal{C}_k(S)$ and $\mathcal{T} = \{T_1, T_2, \ldots, T_l\} \in \mathcal{C}_l(S)$ of $S$, we can create a new clustering $\mathcal{U} = \{U_1, U_2, \ldots, U_m\} \in \mathcal{C}_m(S)$ to be their common refinement. That is, the clusters of $\mathcal{U}$ consist of those elements that are in the same clusters of both $\mathcal{S}$ and $\mathcal{T}$. Formally,

$$\mathcal{U} = \{\, S_i \cap T_j : 1 \leq i \leq k, \, 1 \leq j \leq l \,\}$$

**Lemma 5.** *Let* $\mathcal{S} = \{S_1, S_2, \ldots, S_k\} \in \mathcal{C}_k(S)$ *and* $\mathcal{T} = \{T_1, T_2, \ldots, T_l\} \in \mathcal{C}_l(S)$ *be any two partitions. Let* $\mathcal{U} = \{U_1, U_2, \ldots, U_m\} \in \mathcal{C}_m(S)$ *be their common refinement. Then* $\mathsf{seperation}(\mathcal{U}) = \min\left(\mathsf{seperation}(\mathcal{S}), \mathsf{seperation}(\mathcal{T})\right)$.

*Proof.* It is clear that $\mathsf{seperation}(\mathcal{U}) \leq \min\left(\mathsf{seperation}(\mathcal{S}), \mathsf{seperation}(\mathcal{T})\right)$. To show equality, note that if $a, b$ are in different clusters of $\mathcal{U}$, then $a, b$ must have been in different clusters of either $\mathcal{S}$ or $\mathcal{T}$. $\qquad\qquad\square$

This result can be thought of as expressing a relationship between seperation and the lattice of partitions of $S$ which will be important to our later robustness extension

**Lemma 6.** *Suppose that* $\mathcal{S} = \{S_1, S_2\} \in \mathcal{O}_2(S)$ *is an optimal 2-clustering. Then there is always an optimal $k$-clustering that is a refinement of* $\mathcal{S}$.

*Proof.* Suppose that this is not the case. If $\mathcal{T} = \{T_1, T_2, \ldots, T_k\} \in \mathcal{O}_k(S)$ is an optimal $k$-clustering, let $r$ be the number of clusters of $\mathcal{T}$ that "do not respect" the partition $\{S_1, S_2\}$. That is, $r$ is the number of clusters of $\mathcal{T}$ that intersect both $S_1$ and $S_2 : r = |\{1 \leq i \leq k : T_i \cap S_1 \neq \phi \text{ and } T_i \cap S_2 \neq \phi\}|$. Pick $\mathcal{T} \in \mathcal{O}_k(S)$ to have the smallest $r$. If $r = 0$, then $\mathcal{T}$ is a refinement of $\mathcal{S}$ and there is nothing to show. Otherwise, $r \geq 1$. Assume WLOG that $T_1^{(1)} = T_1 \cap S_1 \neq \phi$ and $T_1^{(2)} = T_1 \cap S_2 \neq \phi$. Then $\mathcal{T}' = \left\{T_1^{(1)}, T_1^{(2)}, T_2, T_3, \ldots, T_k\right\} \in \mathcal{C}_{k+1}(S)$ is a refinement of $\mathcal{T}$ and satisfies $\mathsf{seperation}(\mathcal{T}') = \mathsf{seperation}(\mathcal{T})$. This follows from Lemma 3 along with the fact that **(1)** $D(T_i, T_j) \geq \mathsf{seperation}(\mathcal{T})$ for any $2 \leq i < j \leq k$, **(2)** $D(T_1^{(i)}, T_j) \geq \mathsf{seperation}(\mathcal{T})$ for any $i \in \{1, 2\}$ and $2 \leq j \leq k$, **(3)** $D(T_1^{(1)}, T_1^{(2)}) \geq \mathsf{seperation}(\{S_1, S_2\}) = v_2(S) \geq v_k(S) = \mathsf{seperation}(\mathcal{T})$.

Now, pick two clusters of $\mathcal{T}'$ that are either both contained in the same cluster of $\mathcal{S}$ or both "do not respect" $\mathcal{S}$. Clearly this can always be done. Merge these clusters together to get an element $\mathcal{T}'' \in \mathcal{C}_k(S)$. By Lemma 3 merging clusters cannot decrease the margin. Therefore, $\mathsf{seperation}(\mathcal{T}'') = \mathsf{seperation}(\mathcal{T}') = \mathsf{seperation}(\mathcal{T})$. However, $\mathcal{T}''$ has fewer clusters that do not respect $\mathcal{S}$ hand $\mathcal{T}$ has, and hence we have a contradiction. $\qquad\square$

This lemma implies that Queyranne's algorithm, along with a simple dynamic programming algorithm can be used to find the best $k$ clustering with time complexity $O(k\,|S|^3)$. Observe that in fact this problem can be solved in time $O(|S|^2)$ ([8]). Even though using Queyranne's algorithm is not the fastest algorithm for this problem, the fact that it optimizes this criterion implies that it can be used to optimize conic combinations of submodular criteria and the single-linkage criterion.

### 3.3 Generating robust clusterings

One possible issue with the metric we defined is that it is very sensitive to outliers and noise. To see this, note that if we have two very well separated clusters, then adding a few points "between" the clusters could dramatically decrease the separation. To increase the robustness of the algorithm, we can try to maximize the $n$ smallest distances instead of maximizing just the smallest distance between clusters. If we give the $n$th smallest distance more importance than the smallest distance, this increases the noise tolerance by

ignoring the effects of a few outliers. We will take $n \in \mathbb{N}$ to be some fixed positive integer specified by the user. This will represent the desired degree of noise tolerance (larger gives more noise tolerance). Let $\mathcal{R}_n$ be the set of decreasing $n$-tuples of elements in $\mathbb{R} \cup \{\infty\}$. Given disjoint sets $R, T \subseteq S$, let $D(R, T)$ be the element of $\mathcal{R}_n$ obtained as follows. Let $L(R, T) = \langle d_1, d_2, \ldots, d_{|R| \cdot |T|} \rangle$ be an ordered list of distances between elements of $R$ and $T$ arranged in decreasing order. So for example, if $R = \{1, 2\}$ and $T = \{3, 4\}$, with $d(r, t) = r \cdot t$, then $L(R, T) = \langle 8, 6, 4, 3 \rangle$. We define $D(R, T)$ as follows. If $|R| \cdot |T| \geq n$, then $D(R, T)$ is the last (and thus least) $n$ elements of $L(R, T)$. Otherwise, if $|R| \cdot |T| < n$, then the first $n - |R| \cdot |T|$ elements of $D(R, T)$ are $\infty$, while the remaining elements are the elements of $L(R, T)$. So for example, if $n = 2$, then $D(R, T)$ in the above example would be $\langle 4, 3 \rangle$, if $n = 3$ then $D(R, T) = \langle 6, 4, 3 \rangle$ and if $n = 6$, then $D(R, T) = \langle \infty, \infty, 8, 6, 4, 3 \rangle$.

We define an operation $\oplus$ on $\mathcal{R}_n$ as follows. To get $\langle l_1, l_2, \ldots, l_n \rangle \oplus \langle r_1, r_2, \ldots, r_n \rangle$, order the elements of $\langle l_1, l_2, \ldots, l_n, r_1, r_2, \ldots, r_n \rangle$ in decreasing order, and let $\langle s_1, s_2, \ldots, s_n \rangle$ be the last $n$ elements. For example, $\langle \infty, 3, 2 \rangle \oplus \langle \infty, 6, 5 \rangle = \langle 5, 3, 2 \rangle$ and $\langle 4, 3, 1 \rangle \oplus \langle 5, 4, 3 \rangle = \langle 3, 3, 1 \rangle$. So, the $\oplus$ operation picks off the $n$ smallest elements. It is clear that this operation is commutative (symmetric), associative and that $\langle \infty, \infty, \ldots, \infty \rangle$ acts as an identity. Therefore, $\mathcal{R}_n$ forms a commutative semigroup. In fact, we can describe $D(R, T)$ as follows. For any pair of distinct elements $r, t \in S$, let $d'(r, t) = \langle \infty, \infty, \ldots, d(r, t) \rangle$. Then $D(R, T) = \bigoplus_{r \in R, t \in T} d'(r, t)$. Notice the similarity to $D(R, T) = \min_{r \in R, t \in T} d(r, t)$. In fact, if we take $n = 1$, then the $\oplus$ operation reduces to the minimum operation and we get back our original definitions. We can order $\mathcal{R}_n$ lexicographically. Therefore, $\mathcal{R}_n$ becomes an ordered semigroup. It is entirely straightforward to check that if $R \subseteq T$, then $D(U, T) \prec D(U, R)$, and that if $A, B, W$ are disjoint sets with $D(A, W) \prec D(B, W)$, then $D(A, W \cup B) \prec D(B, A \cup W)$. It is also straightforward to extend Rizzi's proof to see that Queyranne's algorithm (with the obvious modifications) will generate a 2-clustering that minimizes this metric. It can also be verified that the results of Section 3.2 can be extended to this framework (also with the obvious modifications).

In our experiments, we observed that selecting the parameter $n$ is quite tricky. Now, Queyranne's algorithm actually produces a (Gomory-Hu) tree [1] whose edges represent the cost of separating elements. In practice we noticed that restricting our search to only edges whose deletion results in clusters of at least certain sizes produces very good results. Other heuristics such as running the algorithm a number of times to eliminate outliers are also reasonable approaches. Modifying the algorithm to yield good results while retaining the theoretical guarantees is an open question.

## 4  MDL Clustering

We assume that $S$ is a collection of random variables for which we have a (generative) probability model. Since we have the joint probabilities of all subsets of the random variables, the entropy of any collection of the variables is well defined. The expected coding (or description) length of any collection $T$ of random variables using an optimal coding scheme (or a random coding scheme) is known to be $H(T)$. The partition $\{S_1, S_2\}$ of $S$ that minimizes the coding length is therefore $\arg\min_{\{S_1, S_2\} \in \mathcal{C}_2(S)} H(S_1) + H(S_2)$. Now,

$$\arg\min_{\{S_1, S_2\} \in \mathcal{C}_2(S)} H(S_1) + H(S_2) = \arg\min_{\{S_1, S_2\} \in \mathcal{C}_2(S)} H(S_1) + H(S_2) - H(S)$$
$$= \arg\min_{\{S_1, S_2\} \in \mathcal{C}_2(S)} I(S_1; S_2)$$

where $I(S_1; S_2)$ is the mutual information between $S_1$ and $S_2$ because $S_1 \cup S_2 = S$ for all $\{S_1, S_2\} \in \mathcal{C}_2(S)$, Therefore, the problem of partitioning $S$ into two parts to minimize the description length is equivalent to partitioning $S$ into two parts to minimize the mutual information between the parts. It is shown in [9] that the function $f : 2^S \to \mathbb{R}$

defined by $f(T) = I(T; S \setminus T)$ is symmetric and submodular. Clearly the minima of this function correspond to partitions that minimize the mutual information between the parts. Therefore, the problem of partitioning in order to minimize the mutual information between the parts can be reduced to a symmetric submodular minimization problem, which can be solved using Queyranne's algorithm in time $O(|S|^3)$ assuming oracle queries to a mutual information oracle. While implementing such a mutual information oracle is not trivial, for many realistic applications (including one we consider in this paper), the cost of computing a mutual information query is bounded above by the size of the data set, and so the entire algorithm is polynomial in the size of the data set. Symmetric submodular functions generalize notions like graph-cuts, and indeed, Queyranne's algorithm generalizes an algorithm for computing graph-cuts. Since graph-cut based techniques are extensively used in many engineering applications, it might be possible to develop criteria that are more appropriate for these specific applications, while still retaining producing optimal partitions of size 2.

It should be noted that, in general, we cannot use the dynamic programming algorithm to produce optimal clusterings with $k > 2$ clusters for the MDL criterion (or for general symmetric submodular functions). The key reason is that we cannot prove the equivalent of Lemma 6 for the MDL criterion. However, such an algorithm seems reasonable, and it does produce reasonable results. Another approach (which is computationally cheaper) is to compute $k$ clusters by deleting $k - 1$ edges of the Gomory-Hu tree produced by Queyranne's algorithm. It can be shown [9] that this will yield a factor 2 approximation to the optimal $k$-clustering. More generally, if we have an arbitrary increasing submodular function (such as entropy) $f : 2^S \to \mathbb{R}$, and we seek a clustering $\{S_1, S_2, \ldots, S_k\}$ to minimize the sum $\sum_{i=1}^{k} f(S_i)$, then we have an exact algorithm for 2-clusterings and a factor 2 approximation guarantee. Therefore, this generalizes approximation guarantees for graph $k$-cuts because for any graph $G = (V, E)$, the function $f : 2^V \to \mathbb{R}$ where $f(A)$ is the number of edges adjacent to the vertex set $A$ is a submodular function. The finding a clustering to minimize $\sum_{i=1}^{k} f(S_i)$ is equivalent to finding a partition of the vertex set of size $k$ to minimize the number of edges disconnected (i.e., to the graph $k$-cut problem). Another criterion which we can define similarly can be applied to clustering genomic sequences. Intuitively, two genomes are more closely related if they share more common subsequences. Therefore, a natural clustering criterion for sequences is to partition the sequences into clusters so that the sequences from different clusters share as few subsequences as possible. This problem too can be solved using this generic framework.

## 5   Results

Table 1 compares Q-Clustering with various other algorithms. The left part of the table shows the error rates (in percentages) of the (robust) single-linkage criterion and some other techniques on the same data set as is reported in [3]. The data sets are images (of digits and faces), and the distance function we used was the Euclidean distance between the vector of the pixels in the images. The right part of the table compares the Q-Clustering using MDL criterion with other state of the art algorithms for haplotype tagging of SNPs (single nucleotide polymorphisms) in the ACE gene on the data set reported in [4]. In this problem, the goal is to identify a set of SNPs that can accurately predict at least 90% of the SNPs in ACE gene. Typically the SNPs are highly correlated, and so it is necessary to cluster SNPs to identify the correlated SNPs. Note it is very important to identify as few SNPs as possible because the number of clinical trials required grows exponentially with the number of SNPs. As can be seen Q-Clustering does very well on this data set.

## 6   Conclusions

The maximum-separation (single-linkage) metric is a very natural "discriminative" criterion, and it has several advantages, including insensitivity to any monotone transformation of the distances. However, it is quite sensitive to outliers. The robust version does help

|                        | Error rate on Digits | Error rate on Faces |
|------------------------|:--------------------:|:-------------------:|
| Q-Clustering           | 1.4                  | 0                   |
| Max-Margin[†]          | 3                    | 0                   |
| Spectral Clust.[†]     | 6                    | 16.7                |
| K-means[†]             | 7                    | 24.4                |

|                        | #SNPs required |
|------------------------|:--------------:|
| Q-Clustering           | 3              |
| EigenSNP[‡]            | 5              |
| Sliding Window[‡]      | 15             |
| htStep (up)[‡]         | 7              |
| htStep (down)[‡]       | 7              |

Table 1: Comparing (robust) max-separation and MDL Q-Clustering with other techniques. Results marked by [†] and [‡] are from [3] and [4] respectively.

a little, but it does require some additional knowledge (about the approximate number of outliers) and considerable tuning. It is possible that we could develop additional heuristics to automatically determine the parameters of the robust version. The MDL criterion is also a very natural one, and the results on haplotype tagging are quite promising. The MDL criterion can be seen as a generalization of graph cuts, and so it seems like Q-clustering can also be applied to optimize other criteria arising in problems like image segmentation, especially when there is a generative model. Another natural criterion for clustering strings is to partition the strings/sequences to minimize the number of common subsequences. This could have interesting applications in genomics. The key novelty of this paper is the guarantees of optimality produced by the algorithm, and the generaly framework into which a number of natural criterion fall.

## 7  Acknowledgments

The authors acknowledge the assistance of Linli Xu in obtaining the data to test the algorithm and for providing the code used in [3]. Gilles Blanchard pointed out that the MST algorithm finds the optimal solution for the single-linkage criterion. The first and third authors were supported by NSF grant IIS-0093430 and an Intel Corporation Grant.

## References

[1] M. Queyranne. "Minimizing symmetric submodular functions", *Math. Programming*, 82, pages 3–12. 1998.

[2] R. Rizzi, "On Minimizing symmetric set functions", Combinatorica 20(3), pages 445–450, 2000.

[3] L. Xu, J. Neufeld, B. Larson and D. Schuurmans. "Maximum Margin Clustering", in Advances in Neural Information Processing Systems 17, pages 1537-1544, 2005.

[4] Z. Lin and R. B. Altman. "Finding Haplotype Tagging SNPs by Use of Principal Components Analysis", Am. J. Hum. Genet. 75, pages 850-861, 2004.

[5] Jain, A.K. and R.C. Dubes, "Algorithms for Clustering Data." Englewood Cliffs, N.J.: Prentice Hall, 1988.

[6] P. Brucker, "On the complexity of clustering problems," in R. Henn, B. Korte, and W. Oletti (eds.), Optimization and Operations Research, Lecture Notes in Economics and Mathematical Systems, Springer, Berlin 157.

[7] P. Kontkanen, P. Myllymäki, W. Buntine, J. Rissanen and H. Tirri. "An MDL framework for data clustering", HIIT Technical Report 2004.

[8] M. Delattre and P. Hansen. "Bicriterion Cluster Analysis", IEEE Transactions on Pattern Analysis and Machine Intelligence, Vol-2, No. 4, 1980

[9] M. Narasimhan, N. Jojic and J. Bilmes. "Q-Clustering", Technical Report, Dept. of Electrical Engg., University of Washington, UWEETR-2006-0001, 2005
